# An Homotopy Algorithm for the Lasso with Online Observations

**Pierre J. Garrigues**
Department of EECS
Redwood Center for Theoretical Neuroscience
University of California
Berkeley, CA 94720
garrigue@eecs.berkeley.edu

**Laurent El Ghaoui**
Department of EECS
University of California
Berkeley, CA 94720
elghaoui@eecs.berkeley.edu

## Abstract

It has been shown that the problem of $\ell_1$-penalized least-square regression commonly referred to as the Lasso or Basis Pursuit DeNoising leads to solutions that are sparse and therefore achieves model selection. We propose in this paper RecLasso, an algorithm to solve the Lasso with online (sequential) observations. We introduce an optimization problem that allows us to compute an homotopy from the current solution to the solution after observing a new data point. We compare our method to Lars and Coordinate Descent, and present an application to compressive sensing with sequential observations. Our approach can easily be extended to compute an homotopy from the current solution to the solution that corresponds to removing a data point, which leads to an efficient algorithm for leave-one-out cross-validation. We also propose an algorithm to automatically update the regularization parameter after observing a new data point.

## 1 Introduction

Regularization using the $\ell_1$-norm has attracted a lot of interest in the statistics [1], signal processing [2], and machine learning communities. The $\ell_1$ penalty indeed leads to sparse solutions, which is a desirable property to achieve model selection, data compression, or for obtaining interpretable results. In this paper, we focus on the problem of $\ell_1$-penalized least-square regression commonly referred to as the Lasso [1]. We are given a set of training examples or observations $(y_i, x_i) \in \mathbb{R} \times \mathbb{R}^m$, $i = 1 \dots n$. We wish to fit a linear model to predict the response $y_i$ as a function of $x_i$ and a feature vector $\theta \in \mathbb{R}^m$, $y_i = x_i^T \theta + \nu_i$, where $\nu_i$ represents the noise in the observation. The Lasso optimization problem is given by

$$\min_{\theta} \frac{1}{2} \sum_{i=1}^{n} (x_i^T \theta - y_i)^2 + \mu_n \|\theta\|_1, \tag{1}$$

where $\mu_n$ is a regularization parameter. The solution of (1) is typically sparse, i.e. the solution $\theta$ has few entries that are non-zero, and therefore identifies which dimensions in $x_i$ are useful to predict $y_i$.

The $\ell_1$-regularized least-square problem can be formulated as a convex quadratic problem (QP) with linear equality constraints. The equivalent QP can be solved using standard interior-point methods (IPM) [3] which can handle medium-sized problems. A specialized IPM for large-scale problems was recently introduced in [4]. Homotopy methods have also been applied to the Lasso to compute the full regularization path when $\lambda$ varies [5] [6][7]. They are particularly efficient when the solution is very sparse [8]. Other methods to solve (1) include iterative thresholding algorithms [9][10][11], feature-sign search [12], bound optimization methods [13] and gradient projection algorithms [14].

We propose an algorithm to compute the solution of the Lasso when the training examples $(y_i, x_i)_{i=1...N}$ are obtained sequentially. Let $\theta^{(n)}$ be the solution of the Lasso after observing $n$ training examples and $\theta^{(n+1)}$ the solution after observing a new data point $(y_{n+1}, x_{n+1}) \in \mathbb{R} \times \mathbb{R}^m$. We introduce an optimization problem that allows us to compute an homotopy from $\theta^{(n)}$ to $\theta^{(n+1)}$. Hence we use the previously computed solution as a "warm-start", which makes our method particularly efficient when the supports of $\theta^{(n)}$ and $\theta^{(n+1)}$ are close.

In Section 2 we review the optimality conditions of the Lasso, which we use in Section 3 to derive our algorithm. We test in Section 4 our algorithm numerically, and show applications to compressive sensing with sequential observations and leave-one-out cross-validation. We also propose an algorithm to automatically select the regularization parameter each time we observe a new data point.

## 2 Optimality conditions for the Lasso

The objective function in (1) is convex and non-smooth since the $\ell_1$ norm is not differentiable when $\theta_i = 0$ for some $i$. Hence there is a global minimum at $\theta$ if and only if the subdifferential of the objective function at $\theta$ contains the 0-vector. The subdifferential of the $\ell_1$-norm at $\theta$ is the following set

$$\partial \|\theta\|_1 = \left\{ v \in \mathbb{R}^m \; : \; \begin{cases} v_i = \text{sgn}(\theta_i) \text{ if } |\theta_i| > 0 \\ v_i \in [-1, 1] \text{ if } \theta_i = 0 \end{cases} \right\}.$$

Let $X \in \mathbb{R}^{n \times m}$ be the matrix whose $i^{th}$ row is equal to $x_i^T$, and $y = (y_1, \ldots, y_n)^T$. The optimality conditions for the Lasso are given by

$$X^T(X\theta - y) + \mu_n v = 0, \; v \in \partial\|\theta\|_1.$$

We define as the active set the indices of the elements of $\theta$ that are non-zero. To simplify notations we assume that the active set appears first, i.e. $\theta^T = (\theta_1^T, 0^T)$ and $v^T = (v_1^T, v_2^T)$, where $v_{1i} = \text{sgn}(\theta_{1i})$ for all $i$, and $-1 \leq v_{2j} \leq 1$ for all $j$. Let $X = (X_1 \; X_2)$ be the partitioning of $X$ according to the active set. If the solution is unique it can be shown that $X_1^T X_1$ is invertible, and we can rewrite the optimality conditions as

$$\begin{cases} \theta_1 = (X_1^T X_1)^{-1}(X_1^T y - \mu_n v_1) \\ -\mu_n v_2 = X_2^T(X_1 \theta_1 - y) \end{cases}.$$

Note that if we know the active set and the signs of the coefficients of the solution, then we can compute it in closed form.

## 3 Proposed homotopy algorithm

### 3.1 Outline of the algorithm

Suppose we have computed the solution $\theta^{(n)}$ to the Lasso with $n$ observation and that we are given an additional observation $(y_{n+1}, x_{n+1}) \in \mathbb{R} \times \mathbb{R}^m$. Our goal is to compute the solution $\theta^{(n+1)}$ of the augmented problem. We introduce the following optimization problem

$$\theta(t, \mu) = \arg\min_{\theta} \frac{1}{2} \left\| \begin{pmatrix} X \\ tx_{n+1}^T \end{pmatrix} \theta - \begin{pmatrix} y \\ ty_{n+1} \end{pmatrix} \right\|_2^2 + \mu\|\theta\|_1. \tag{2}$$

We have $\theta^{(n)} = \theta(0, \mu_n)$ and $\theta^{(n+1)} = \theta(1, \mu_{n+1})$. We propose an algorithm that computes a path from $\theta^{(n)}$ to $\theta^{(n+1)}$ in two steps:

**Step 1** Vary the regularization parameter from $\mu_n$ to $\mu_{n+1}$ with $t = 0$. This amounts to computing the regularization path between $\mu_n$ and $\mu_{n+1}$ as done in Lars. The solution path is piecewise linear and we do not review it in this paper (see [15][7][5]).

**Step 2** Vary the parameter $t$ from 0 to 1 with $\mu = \mu_{n+1}$. We show in Section 3.2 how to compute this path.

## 3.2 Algorithm derivation

We show in this Section that $\theta(t, \mu)$ is a piecewise smooth function of $t$. To make notations lighter we write $\theta(t) := \theta(t, \mu)$. We saw in Section 2 that the solution to the Lasso can be easily computed once the active set and signs of the coefficients are known. This information is available at $t = 0$, and we show that the active set and signs will remain the same for $t$ in an interval $[0, t^*)$ where the solution $\theta(t)$ is smooth. We denote such a point where the active set changes a "transition point" and show how to compute it analytically. At $t^*$ we update the active set and signs which will remain valid until $t$ reaches the next transition point. This process is iterated until we know the active set and signs of the solution at $t = 1$, and therefore can compute the desired solution $\theta^{(n+1)}$.

We suppose as in Section 2 and without loss of generality that the solution at $t = 0$ is such that $\theta(0) = (\theta_1^T, 0^T)$ and $v^T = (v_1^T, v_2^T) \in \partial \|\theta(0)\|_1$ satisfy the optimality conditions.

**Lemma 1.** Suppose $\theta_{1i} \neq 0$ for all $i$ and $|v_{2j}| < 1$ for all $j$. There exist $t^* > 0$ such that for all $t \in [0, t^*)$, the solution of (2) has the same support and the same sign as $\theta(0)$.

PROOF. The optimality conditions of (2) are given by

$$X^T (X\theta - y) + t^2 x_{n+1} \left( x_{n+1}^T \theta - y_{n+1} \right) + \mu w = 0, \qquad (3)$$

where $w \in \partial \|\theta\|_1$. We show that there exists a solution $\theta(t)^T = (\theta_1(t)^T, 0^T)$ and $w(t)^T = (v_1^T, w_2(t)^T) \in \partial \|\theta(t)\|_1$ satisfying the optimality conditions for $t$ sufficiently small. We partition $x_{n+1}^T = (x_{n+1,1}^T, x_{n+1,2}^T)$ according to the active set. We rewrite the optimality conditions as

$$\begin{cases} X_1^T (X_1\theta_1(t) - y) + t^2 x_{n+1,1} \left( x_{n+1,1}^T \theta_1(t) - y_{n+1} \right) + \mu v_1 = 0 \\ X_2^T (X_1\theta_1(t) - y) + t^2 x_{n+1,2} \left( x_{n+1,1}^T \theta_1(t) - y_{n+1} \right) + \mu w_2(t) = 0 \end{cases}.$$

Solving for $\theta_1(t)$ using the first equation gives

$$\theta_1(t) = \left( X_1^T X_1 + t^2 x_{n+1,1} x_{n+1,1}^T \right)^{-1} \left( X_1^T y + t^2 y_{n+1} x_{n+1,1} - \mu v_1 \right). \qquad (4)$$

We can see that $\theta_1(t)$ is a continuous function of $t$. Since $\theta_1(0) = \theta_1$ and the elements of $\theta_1$ are all strictly positive, there exists $t_1^*$ such that for $t < t_1^*$, all elements of $\theta_1(t)$ remain positive and do not change signs. We also have

$$-\mu_{n+1} w_2(t) = X_2^T (X_1\theta_1(t) - y) + t^2 x_{n+1,2} \left( x_{n+1,1}^T \theta_1(t) - y_{n+1} \right). \qquad (5)$$

Similarly $w_2(t)$ is a continuous function of $t$, and since $w_2(0) = v_2$, there exists $t_2^*$ such that for $t < t_2^*$ all elements of $w_2(t)$ are strictly smaller than 1 in absolute value. By taking $t^* = \min(t_1^*, t_2^*)$ we obtain the desired result. □

The solution $\theta(t)$ will therefore be smooth until $t$ reaches a transition point where either a component of $\theta_1(t)$ becomes zero, or one of the component of $w_2(t)$ reaches one in absolute value. We now show how to compute the value of the transition point.

Let $\tilde{X} = \begin{pmatrix} X \\ x_{n+1}^T \end{pmatrix}$ and $\tilde{y} = \begin{pmatrix} y \\ y_{n+1} \end{pmatrix}$. We partition $\tilde{X} = (\tilde{X}_1 \ \tilde{X}_2)$ according to the active set. We use the Sherman-Morrison formula and rewrite (4) as

$$\theta_1(t) = \tilde{\theta}_1 - \frac{(t^2 - 1)\bar{e}}{1 + \alpha(t^2 - 1)} u,$$

where $\tilde{\theta}_1 = (\tilde{X}_1^T \tilde{X}_1)^{-1}(\tilde{X}_1^T \tilde{y} - \mu v_1)$, $\bar{e} = x_{n+1,1}^T \tilde{\theta}_1 - y_{n+1}$, $\alpha = x_{n+1,1}^T (\tilde{X}_1^T \tilde{X}_1)^{-1} x_{n+1,1}$ and $u = (\tilde{X}_1^T \tilde{X}_1)^{-1} x_{n+1,1}$. Let $t_{1i}$ the value of $t$ such that $\theta_{1i}(t) = 0$. We have

$$t_{1i} = \left( 1 + \left( \frac{\bar{e} u_i}{\tilde{\theta}_{1i}} - \alpha \right)^{-1} \right)^{\frac{1}{2}},$$

We now examine the case where a component of $w_2(t)$ reaches one in absolute value. We first notice that

$$\begin{cases} x_{n+1,1}^T \theta_1(t) - y_{n+1} = \frac{\bar{e}}{1 + \alpha(t^2 - 1)} \\ \tilde{X}_1 \theta_1(t) - \tilde{y} = \tilde{e} - \frac{(t^2 - 1)\bar{e}}{1 + \alpha(t^2 - 1)} \tilde{X}_1 u \end{cases},$$

where $\tilde{e} = \tilde{X}_1\tilde{\theta}_1 - \tilde{y}$. We can rewrite (5) as

$$-\mu w_2(t) = \tilde{X}_2^T \tilde{e} + \frac{\bar{e}(t^2 - 1)}{1 + \alpha(t^2 - 1)}(x_{n+1,2} - \tilde{X}_2^T \tilde{X}_1 u).$$

Let $c_j$ be the $j^{th}$ column of $\tilde{X}_2$, and $x^{(j)}$ the $j^{th}$ element of $x_{n+1,2}$. The $j^{th}$ component of $w_2(t)$ will become 1 in absolute value as soon as

$$\left| c_j^T \tilde{e} + \frac{\bar{e}(t^2 - 1)}{1 + \alpha(t^2 - 1)}\left( x^{(j)} - c_j^T \tilde{X}_1 u \right) \right| = \mu.$$

Let $t_{2\,j}^+$ (resp. $t_{2\,j}^-$) be the value such that $w_{2j}(t) = 1$ (resp. $w_{2j}(t) = -1$). We have

$$\begin{cases} t_{2\,j}^+ = \left( 1 + \left( \frac{\bar{e}(x^{(j)} - c_j^T \tilde{X}_1 u)}{-\mu - c_j^T \tilde{e}} - \alpha \right)^{-1} \right)^{\frac{1}{2}} \\ t_{2\,j}^- = \left( 1 + \left( \frac{\bar{e}(x^{(j)} - c_j^T \tilde{X}_1 u)}{\mu - c_j^T \tilde{e}} - \alpha \right)^{-1} \right)^{\frac{1}{2}} \end{cases}.$$

Hence the transition point will be equal to $t' = \min\{\min_i t_{1i}, \ \min_j t_{2\,j}^+, \ \min_j t_{2\,j}^-\}$ where we restrict ourselves to the real solutions that lie between 0 and 1. We now have the necessary ingredients to derive the proposed algorithm.

---

**Algorithm 1** RecLasso: homotopy algorithm for online Lasso

---

1: Compute the path from $\theta^{(n)} = \theta(0, \mu_n)$ to $\theta(0, \mu_{n+1})$.
2: Initialize the active set to the non-zero coefficients of $\theta(0, \mu_{n+1})$ and let $v = \text{sign}(\theta(0, \mu_{n+1}))$.
   Let $v_1$ and $x_{n+1,1}$ be the subvectors of $v$ and $x_{n+1}$ corresponding to the active set, and $\tilde{X}_1$ the submatrix of $\tilde{X}$ whose columns correspond to the active set.
   Initialize $\tilde{\theta}_1 = (\tilde{X}_1^T \tilde{X}_1)^{-1}(\tilde{X}_1^T \tilde{y} - \mu v_1)$.
   Initialize the transition point $t' = 0$.
3: Compute the next transition point $t'$. If it is smaller than the previous transition point or greater than 1, go to Step 5.
   **Case 1** The component of $\theta_1(t')$ corresponding to the $i^{th}$ coefficient goes to zero:
        Remove $i$ from the active set.
        Update $v$ by setting $v_i = 0$.
   **Case 2** The component of $w_2(t')$ corresponding to the $j^{th}$ coefficient reaches one in absolute value:
        Add $j$ to the active set.
        If the component reaches 1 (resp. $-1$), then set $v_j = 1$ (resp. $v_j = -1$).
4: Update $v_1$, $\tilde{X}_1$ and $x_{n+1,1}$ according to the updated active set.
   Update $\tilde{\theta}_1 = (\tilde{X}_1^T \tilde{X}_1)^{-1}(\tilde{X}_1^T \tilde{y} - \mu v_1)$ (rank 1 update).
   Go to Step 3.
5: Compute final value at $t = 1$, where the values of $\theta^{(n+1)}$ on the active set are given by $\tilde{\theta}_1$.

---

The initialization amounts to computing the solution of the Lasso when we have only one data point $(y, x) \in \mathbb{R} \times \mathbb{R}^m$. In this case, the active set has at most one element. Let $i_0 = \arg\max_i |x^{(i)}|$ and $v = \text{sign}(yx^{(i_0)})$. We have

$$\theta^{(1)} = \begin{cases} \frac{1}{(x^{(i_0)})^2}(yx^{(i_0)} - \mu_1 v)e_{i_0} & \text{if } |yx^{(i_0)}| > \mu_1 \\ 0 & \text{otherwise} \end{cases}.$$

We illustrate our algorithm by showing the solution path when the regularization parameter and $t$ are successively varied with a simple numerical example in Figure 1.

## 3.3 Complexity

The complexity of our algorithm is dominated by the inversion of the matrix $\tilde{X}_1^T \tilde{X}_1$ at each transition point. The size of this matrix is bounded by $q = \min(n, m)$. As the update to this matrix after a

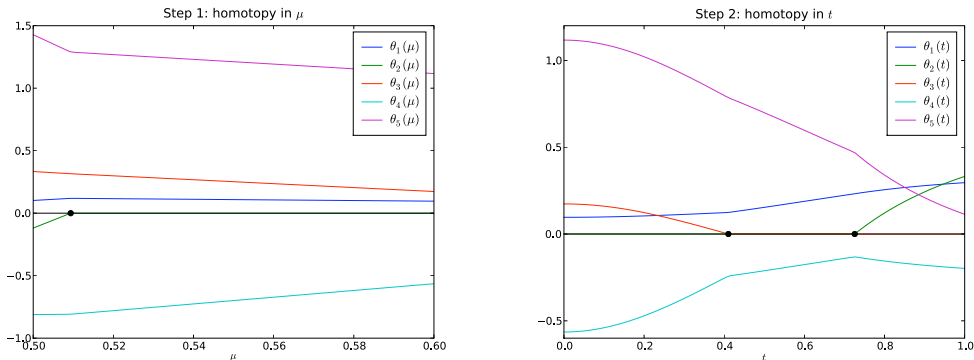

Figure 1: Solution path for both steps of our algorithm. We set $n = 5$, $m = 5$, $\mu_n = .1n$. All the values of $X$, $y$, $x_{n+1}$ and $y_{n+1}$ are drawn at random. On the left is the homotopy when the regularization parameter goes from $\mu_n = .5$ to $\mu_{n+1} = .6$. There is one transition point as $\theta_2$ becomes inactive. On the right is the piecewise smooth path of $\theta(t)$ when $t$ goes from 0 to 1. We can see that $\theta_3$ becomes zero, $\theta_2$ goes from being 0 to being positive, whereas $\theta_1$, $\theta_4$ and $\theta_5$ remain active with their signs unchanged. The three transition points are shown as black dots.

transition point is rank 1, the cost of computing the inverse is $O(q^2)$. Let $k$ be the total number of transition points after varying the regularization parameter from $\mu_n$ to $\mu_{n+1}$ and $t$ from 0 to 1. The complexity of our algorithm is thus $O(kq^2)$. In practice, the size of the active set $d$ is much lower than $q$, and if it remains $\sim d$ throughout the homotopy, the complexity is $O(kd^2)$. It is instructive to compare it with the complexity of recursive least-square, which corresponds to $\mu_n = 0$ for all $n$ and $n > m$. For this problem the solution typically has $m$ non-zero elements, and therefore the cost of updating the solution after a new observation is $O(m^2)$. Hence if the solution is sparse ($d$ small) and the active set does not change much ($k$ small), updating the solution of the Lasso will be faster than updating the solution to the non-penalized least-square problem.

Suppose that we applied Lars directly to the problem with $n + 1$ observations without using knowledge of $\theta^{(n)}$ by varying the regularization parameter from a large value where the size of the active set is 0 to $\mu_{n+1}$. Let $k'$ be the number of transition points. The complexity of this approach is $O(k'q^2)$, and we can therefore compare the efficiency of these two approaches by comparing the number of transition points.

## 4 Applications

### 4.1 Compressive sensing

Let $\theta_0 \in \mathbb{R}^m$ be an unknown vector that we wish to reconstruct. We observe $n$ linear projections $y_i = x_i^T \theta_0 + \nu_i$, where $\nu_i$ is Gaussian noise of variance $\sigma^2$. In general one needs $m$ such measurement to reconstruct $\theta_0$. However, if $\theta_0$ has a sparse representation with $k$ non-zero coefficients, it has been shown in the noiseless case that it is sufficient to use $n \propto k \log m$ such measurements. This approach is known as compressive sensing [16][17] and has generated a tremendous amount of interest in the signal processing community. The reconstruction is given by the solution of the Basis Pursuit (BP) problem

$$\min_{\theta} \|\theta\|_1 \text{ subject to } X\theta = y.$$

If measurements are obtained sequentially, it is advantageous to start estimating the unknown sparse signal as measurements arrive, as opposed to waiting for a specified number of measurements. Algorithms to solve BP with sequential measurements have been proposed in [18][19], and it has been shown that the change in the active set gives a criterion for how many measurements are needed to recover the underlying signal [19].

In the case where the measurements are noisy ($\sigma > 0$), a standard approach to recover $\theta_0$ is to solve the Basis Pursuit DeNoising problem instead [20]. Hence, our algorithm is well suited for

compressive sensing with sequential and noisy measurements. We compare our proposed algorithm to Lars as applied to the entire dataset each time we receive a new measurement. We also compare our method to coordinate descent [11] with warm start: when receiving a new measurement, we initialize coordinate descent (CD) to the actual solution.

We sample measurements of a model where $m = 100$, the vector $\theta_0$ used to sample the data has 25 non-zero elements whose values are Bernoulli $\pm 1$, $x_i \sim \mathcal{N}(0, I_m)$, $\sigma = 1$, and we set $\mu_n = .1n$. The reconstruction error decreases as the number of measurements grows (not plotted). The parameter that controls the complexity of Lars and RecLasso is the number of transition points. We see in Figure 2 that this quantity is consistently smaller for RecLasso, and that after 100 measurements when the support of the solution does not change much there are typically less than 5 transition points for RecLasso. We also show in Figure 2 timing comparison for the three algorithms that we have each implemented in Python. We observed that CD requires a lot of iterations to converge to the optimal solution when $n < m$, and we found difficult to set a stopping criterion that ensures convergence. Our algorithm is consistently faster than Lars and CD with warm-start.

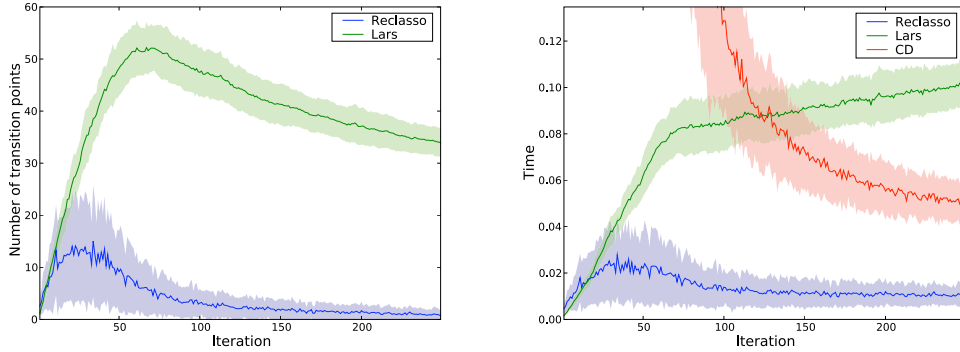

Figure 2: Compressive sensing results. On the x-axis of the plots are the iterations of the algorithm, where at each iteration we receive a new measurement. On the left is the comparison of the number of transition points for Lars and RecLasso, and on the right is the timing comparison for the three algorithms. The simulation is repeated 100 times and shaded areas represent one standard deviation.

### 4.2   Selection of the regularization parameter

We have supposed until now a pre-determined regularization schedule, an assumption that is not practical. The amount of regularization depends indeed on the variance of the noise present in the data which is not known a priori. It is therefore not obvious how to determine the amount of regularization. We write $\mu_n = n\lambda_n$ such that $\lambda_n$ is the weighting factor between the average mean-squared error and the $\ell_1$-norm. We propose an algorithm that selects $\lambda_n$ in a data-driven manner. The problem with $n$ observations is given by

$$\theta(\lambda) = \arg\min_{\theta} \frac{1}{2n} \sum_{i=1}^{n} (x_i^T \theta - y_i)^2 + \lambda \|\theta\|_1.$$

We have seen previously that $\theta(\lambda)$ is piecewise linear, and we can therefore compute its gradient unless $\lambda$ is a transition point. Let $err(\lambda) = (x_{n+1}^T \theta(\lambda) - y_{n+1})^2$ be the error on the new observation. We propose the following update rule to select $\lambda_{n+1}$

$$\log \lambda_{n+1} = \log \lambda_n - \eta \frac{\partial err}{\partial \log \lambda}(\lambda_n)$$

$$\Rightarrow \lambda_{n+1} = \lambda_n \times exp\Big\{2n\eta x_{n+1,1}^T (X_1^T X_1)^{-1} v_1 (x_{n+1}^T \theta_1 - y_{n+1})\Big\},$$

where the solution after $n$ observations corresponding to the regularization parameter $\lambda_n$ is given by $(\theta_1^T, 0^T)$, and $v_1 = \text{sign}(\theta_1)$. We therefore use the new observation as a test set, which allows us to update the regularization parameter before introducing the new observation by varying $t$ from 0

to 1. We perform the update in the $\log$ domain to ensure that $\lambda_n$ is always positive. We performed simulations using the same experimental setup as in Section 4.1 and using $\eta = .01$. We show in Figure 3 a representative example where $\lambda$ converges. We compared this value to the one we would obtain if we had a training and a test set with $250$ observations each such that we could fit the model on the training set for various values of $\lambda$, and see which one gives the smallest prediction error on the test set. We obtain a very similar result, and understanding the convergence properties of our proposed update rule for the regularization parameter is the object of current research.

### 4.3 Leave-one-out cross-validation

We suppose in this Section that we have access to a dataset $(y_i, x_i)_{i=1...n}$ and that $\mu_n = n\lambda$. The parameter $\lambda$ is tied to the amount of noise in the data which we do not know a priori. A standard approach to select this parameter is leave-one-out cross-validation. For a range of values of $\lambda$, we use $n-1$ data points to solve the Lasso with regularization parameter $(n-1)\lambda$ and then compute the prediction error on the data point that was left out. This is repeated $n$ times such that each data point serves as the test set. Hence the best value for $\lambda$ is the one that leads to the smallest mean prediction error.

Our proposed algorithm can be adapted to the case where we wish to update the solution of the Lasso after a data point is removed. To do so, we compute the first homotopy by varying the regularization parameter from $n\lambda$ to $(n-1)\lambda$. We then compute the second homotopy by varying $t$ from 1 to 0 which has the effect of removing the data point that will be used for testing. As the algorithm is very similar to the one we proposed in Section 3.2 we omit the derivation. We sample a model with $n = 32$ and $m = 32$. The vector $\theta_0$ used to generate the data has 8 non-zero elements. We add Gaussian noise of variance $0.2$ to the observations, and select $\lambda$ for a range of 10 values. We show in Figure 4 the histogram of the number of transition points for our algorithm when solving the Lasso with $n-1$ data points (we solve this problem $10 \times n$ times). Note that in the majority cases there are very few transition points, which makes our approach very efficient in this setting.

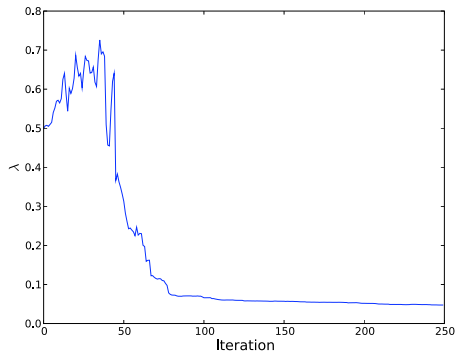

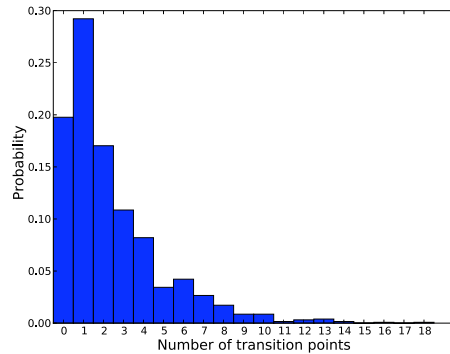

Figure 3: Evolution of the regularization parameter when using our proposed update rule.

Figure 4: Histogram of the number of transition points when removing an observation.

## 5 Conclusion

We have presented an algorithm to solve $\ell_1$-penalized least-square regression with online observations. We use the current solution as a "warm-start" and introduce an optimization problem that allows us to compute an homotopy from the current solution to the solution after observing a new data point. The algorithm is particularly efficient if the active set does not change much, and we show a computational advantage as compared to Lars and Coordinate Descent with warm-start for applications such as compressive sensing with sequential observations and leave-one-out cross-validation. We have also proposed an algorithm to automatically select the regularization parameter where each new measurement is used as a test set.

**Acknowledgments**

We wish to acknowledge support from NSF grant 0835531, and Guillaume Obozinski and Chris Rozell for fruitful discussions.

# References

[1] R. Tibshirani. Regression shrinkage and selection via the lasso. *Journal of the Royal Statistical Society. Series B*, 58(1):267–288, 1996.

[2] S. Chen, D. Donoho, and M. Saunders. Atomic decomposition by basis pursuit. *SIAM Review*, 43(1):129–159, 2001.

[3] S. Boyd and L. Vandenberghe. Convex optimization. *Cambridge Univ. Press*, 2004.

[4] S-J. Kim, K. Koh, M. Lustig, S. Boyd, and D. Gorinevsky. An interior-point method for large-scale l1-regularized least squares. *IEEE Journal of Selected Topics in Signal Processing*, 1(4):606–617, 2007.

[5] B. Efron, T. Hastie, I. Johnstone, and R. Tibshirani. Least angle regression. *Annals of Statistics*, 32(2):407–499, 2004.

[6] M.R. Osborne, B. Presnell, and B.A. Turlach. A new approach to variable selection in least squares problems. *IMA Journal of Numerical Analysis*, 20:389–404, 2000.

[7] D.M. Malioutov, M. Cetin, and A.S. Willsky. Homotopy continuation for sparse signal representation. In *Proceedings of the International Conference on Acoustics, Speech, and Signal Processing (ICASSP)*, Philadelphia, PA, March 2005.

[8] I. Drori and D.L. Donoho. Solution of $\ell_1$ minimization problems by lars/homotopy methods. In *Proceedings of the International Conference on Acoustics, Speech, and Signal Processing (ICASSP)*, Toulouse, France, May 2006.

[9] I. Daubechies, M. Defrise, and C. De Mol. An iterative thresholding algorithm for linear inverse problems with a sparsity constraint. *Communications on Pure and Applied Mathematics*, 57:1413–1541, 2004.

[10] C.J. Rozell, D.H. Johnson, R.G. Baraniuk, and B.A. Olshausen. Locally competitive algorithms for sparse approximation. In *Proceedings of the International Conference on Image Processing (ICIP)*, San Antonio, TX, September 2007.

[11] J. Friedman, T. Hastie, H. Hoefling, and R. Tibshirani. Pathwise coordinate optimization. *The Annals of Applied Statistics*, 1(2):302–332, 2007.

[12] H. Lee, A. Battle, R. Raina, and A.Y. Ng. Efficient sparse coding algorithms. In *Proceedings of the Neural Information Processing Systems (NIPS)*, 2007.

[13] M. Figueiredo and R. Nowak. A bound optimization approach to wavelet-based image deconvolution. In *Proceedings of the International Conference on Image Processing (ICIP)*, Genova, Italy, September 2005.

[14] M. Figueiredo, R. Nowak, and S. Wright. Gradient projection for sparse reconstruction: Application to compressed sensing and other inverse problems. *IEEE Journal of Selected Topics in Signal Processing*, 1(4):586–597, 2007.

[15] M Osborne. An effective method for computing regression quantiles. *IMA Journal of Numerical Analysis*, Jan 1992.

[16] E. Candès. Compressive sampling. *Proceedings of the International Congress of Mathematicians*, 2006.

[17] D.L. Donoho. Compressed sensing. *IEEE Transactions on Information Theory*, 52(4):1289–1306, 2006.

[18] S. Sra and J.A. Tropp. Row-action methods for compressed sensing. In *Proceedings of the International Conference on Acoustics, Speech, and Signal Processing (ICASSP)*, Toulouse, France, May 2006.

[19] D. Malioutov, S. Sanghavi, and A. Willsky. Compressed sensing with sequential observations. In *Proceedings of the International Conference on Acoustics, Speech, and Signal Processing (ICASSP)*, Las Vegas, NV, March 2008.

[20] Y. Tsaig and D.L. Donoho. Extensions of compressed sensing. *Signal Processing*, 86(3):549–571, 2006.

